# Using the Nyström Method to Speed Up Kernel Machines

**Christopher K. I. Williams and Matthias Seeger**
Institute for Adaptive and Neural Computation
University of Edinburgh
5 Forrest Hill, Edinburgh EH1 2QL
*c.k.i.williams@ed.ac.uk, seeger@dai.ed.ac.uk*
*http://anc.ed.ac.uk*

## Abstract

A major problem for kernel-based predictors (such as Support Vector Machines and Gaussian processes) is that the amount of computation required to find the solution scales as $O(n^3)$, where $n$ is the number of training examples. We show that an approximation to the eigendecomposition of the Gram matrix can be computed by the Nyström method (which is used for the numerical solution of eigenproblems). This is achieved by carrying out an eigendecomposition on a smaller system of size $m < n$, and then expanding the results back up to $n$ dimensions. The computational complexity of a predictor using this approximation is $O(m^2 n)$. We report experiments on the USPS and abalone data sets and show that we can set $m \ll n$ without any significant decrease in the accuracy of the solution.

In recent years much attention has been paid to kernel-based classifiers such as Support Vector Machines (SVMs) (Vapnik, 1995), Gaussian process classifiers (e.g. see Williams & Barber, 1998) and spline methods (Wahba, 1990). One of the main drawbacks of kernel-based classifiers is that the computational complexity required to find the solution scales as $O(n^3)$, where $n$ is the number of training examples. In this paper we present a reduced-rank approximation to the Gram matrix $K$, giving rise to $O(m^2 n)$ computational complexity. This approximation $\tilde{K}$ is obtained by randomly choosing $m$ rows/columns of $K$ (without replacement), and then setting $\tilde{K} = K_{n,m} K_{m,m}^{-1} K_{m,n}$, where $K_{n,m}$ is the $n \times m$ block of the original matrix $K$, and with similar definitions for the other blocks. We find that in practice we can set $m \ll n$ without any significant decrease in the accuracy of the solution.

In section 1 of the paper we discuss the theory of the method. Section 2 gives experimental results, and we conclude with a discussion in section 3.

## 1 Theory of the Nyström method

### 1.1 The Nyström method for approximating eigenfunctions

In the theory of kernel machines we consider covariance kernels $k(x, y)$. These can be related to an expansion into a feature space of dimension $N$ (typically $N$ is larger

than the dimension of the input space $x$) so that

$$k(x, y) = \sum_{i=1}^{N} \lambda_i \phi_i(x) \phi_i(y), \qquad (1)$$

where $N \leq \infty$, $\lambda_1 \geq \lambda_2 \geq \cdots \geq 0$ denotes the eigenvalues and $\phi_1, \phi_2, \ldots$ denotes the eigenfunctions of the operator whose kernel is $k$, so that

$$\int k(y, x) \phi_i(x) p(x) dx = \lambda_i \phi_i(y), \qquad (2)$$

where $p(x)$ denotes the probability density of the input vector $x$. The eigenfunctions are $p$-orthogonal, so that $\int \phi_i(x) \phi_j(x) p(x) dx = \delta_{ij}$. To approximate this eigenfunction equation given an iid sample $\{x_1, \ldots, x_q\}$ from $p(x)$, we replace the integral over $p(x)$ by an empirical average to obtain

$$\frac{1}{q} \sum_{k=1}^{q} k(y, x_k) \phi_i(x_k) \approx \lambda_i \phi_i(y). \qquad (3)$$

The $p$-orthogonality of the eigenfunctions translates into the empirical constraint $\frac{1}{q} \sum_{k=1}^{q} \phi_i(x_k) \phi_j(x_k) \approx \delta_{i,j}$. Equation 3 motivates the matrix eigenproblem

$$K^{(q)} U^{(q)} = U^{(q)} \Lambda^{(q)}, \qquad (4)$$

where $K^{(q)}$ is the $q \times q$ Gram matrix with elements $K_{ij}^{(q)} = K(x_i, x_j)$ for $i, j = 1, \ldots, q$, $U^{(q)} \in \mathbb{R}^{q \times q}$ is column orthonormal and $\Lambda^{(q)}$ is a diagonal matrix with entries $\lambda_1^{(q)} \geq \lambda_2^{(q)} \geq \ldots \lambda_q^{(q)} \geq 0$. If we plug the $x_j$ for $y$ into equation 3 and match this against equation 4 we arrive at the following approximations:

$$\phi_i(x_j) \approx \sqrt{q} U_{j,i}^{(q)}, \qquad \lambda_i \approx \frac{\lambda_i^{(q)}}{q}. \qquad (5)$$

Plugging these back in into into equation (3) we obtain the Nyström approximation to the $i$th eigenfunction (see, e.g., Baker, 1977, chapter 3)

$$\phi_i(y) \approx \frac{\sqrt{q}}{\lambda_i^{(q)}} \sum_{k=1}^{q} k(y, x_k) U_{k,i}^{(q)} = \frac{\sqrt{q}}{\lambda_i^{(q)}} k_y \cdot u_i^{(q)}, \qquad (6)$$

where $k_y$ is the vector $(k(x_1, y), \ldots, k(x_q, y))^T$ and $u_i^{(q)}$ is the $i$th column of $U^{(q)}$. Note that equation 6 is identical (up to scaling factors) to equation 4.1 in Schölkopf et al. (1998) which describes the projection of a new point $x$ onto the $i$th eigenvector in feature space.

## 1.2 Using the Nyström method to approximate the Gram matrix

To avoid numerical instabilities due to ill-conditioning it is common practice to replace the Gram matrix $K$ by $K + \sigma I$ (Neal, 1998) where $\sigma$ is a small positive constant called *jitter factor*. One method to cut down on computational costs is to use the eigendecomposition of $K$

$$K = U_F \Lambda_F U_F^T \qquad (7)$$

where $U_F$ is orthonormal, $\Lambda_F = \text{diag}(\lambda_i^{(F)})_i$, $\lambda_1^{(F)} \geq \lambda_2^{(F)} \geq \cdots \geq 0$. Now, for some $p < n$ build $U \in \mathbb{R}^{n \times p}$ from the first $p$ columns of $U_F$ and let

$\Lambda = \operatorname{diag}(\lambda_1^{(F)}, \ldots, \lambda_p^{(F)})$. We can then approximate $K + \sigma I$ by $U \Lambda U^T + \sigma I$. Approximations of this kind are widely used, e.g. in *principal component analysis (PCA)* and can be motivated in a variety of ways.

If the eigendecomposition is available, using the approximation $U \Lambda U^T + \sigma I$ will greatly reduce the computational costs of many kernel methods. Note that this matrix is not singular due to the $\sigma I$ term. However, computing the eigendecomposition is a $O(n^3)$ operation. There are methods to compute the first $p$ eigenvalues and eigenvectors of $K$, but their average running times are significantly below $O(n^3)$ only if $p \ll n$.

However, the Nyström technique described above can be used to compute an approximation to the eigenvalues and eigenvectors we require. If we use a subset of the training data of size $q = m < n$ to create the matrix eigenproblem of equation 7, we can then approximate the eigenfunctions at all $n$ points using equation 5. Let this low-rank approximation to $K$ be denoted by $\tilde{K} = \tilde{U} \tilde{\Lambda} \tilde{U}^T = \sum_{i=1}^{p} \tilde{\lambda}_i^{(n)} \tilde{u}_i^{(n)} (\tilde{u}_i^{(n)})^T$, where $\tilde{\lambda}_i^{(n)}$ and $\tilde{u}_i^{(n)}$ are the Nyström approximations of the eigenvalues/vectors $\lambda_i^{(n)}$ and $u_i^{(n)}$ of the $n \times n$ matrix. By applying equations 5 and 6 with $p \leq m < n$ (and noting that $\lambda_i^{(F)} = \lambda_i^{(n)}$, $i = 1, \ldots, p$ and the first $p$ columns of $U$ and $U^{(n)}$ coincide), we arrive at the approximation formulae:

$$\tilde{\lambda}_i^{(n)} \overset{def}{=} \frac{n}{m} \lambda_i^{(m)}, \qquad\qquad i = 1, \ldots, p \qquad\qquad (8)$$

$$\tilde{u}_i^{(n)} \overset{def}{=} \sqrt{\frac{m}{n}} \frac{1}{\lambda_i^{(m)}} K_{n,m} u_i^{(m)} \qquad\qquad i = 1, \ldots, p \qquad\qquad (9)$$

where $u_i^{(m)}$ is the $i$th eigenvector of the $m \times m$ eigenproblem and $K_{n,m}$ is the appropriate $n \times m$ submatrix of $K$. Note that the entries of $\tilde{u}_i^{(n)}$ at the $m$ points are just rescaled versions of $u_i^{(m)}$. We can therefore compute an approximation to the $U \Lambda U^T + \sigma I$ in time $O(m^3 + pmn) = O(m^2 n)$ using the Nyström technique.

**The nature of the approximation of $K$ for $p = m$.** We consider the quality of the approximation $\tilde{K}$ for $K$ at (i) the $m$ points used for the eigendecomposition and (ii) the $n - m$ other points. Let $K$ be partitioned into blocks $K_{m,m}, K_{n-m,m} = K_{m,n-m}^T$ and $K_{n-m,n-m}$. Plugging the approximation (9) into $\tilde{K} = \tilde{U} \tilde{\Lambda} \tilde{U}^T$ it is easy to show that

$$\tilde{K} = K_{n,m} K_{m,m}^{-1} K_{m,n}. \qquad\qquad (10)$$

Further we see that $K_{m,m} = \tilde{K}_{m,m}$, $K_{m,n-m} = \tilde{K}_{m,n-m}$, $K_{n-m,m} = \tilde{K}_{n-m,m}$, but that $\tilde{K}_{n-m,n-m} = K_{n-m,m} K_{m,m}^{-1} K_{m,n-m}$. The difference $K_{n-m,n-m} - \tilde{K}_{n-m,n-m}$ is in fact the *Schur complement* of $K_{m,m}$. It is easy to show that equation 10 is exact in the case that $K$ has rank $m$ and that $m$ linearly-independent columns are chosen.

## 1.3   Related work

We have recently become aware of the work of Frieze *et al* (1998) who use a weighted random subsampling of the rows and columns of a rectangular matrix to compute an approximation to the SVD of that matrix. As a special case an approximate eigendecomposition of a SPD matrix can be obtained. Our work gives an independent derivation of the result (without the weighting), using Nyström theory, and applies it to kernel machines. Also Smola and Schölkopf (2000) have recently described a sparse greedy matrix approximation method. It turns out that the form of our

approximation is identical to theirs, although they grow the approximate matrix, searching over which row/column to append next, rather than choosing randomly. For a given $m$, the sparse greedy method produces a better approximation to $K$, but at the expense of more computation. Fine and Scheinberg (2000) have also very recently pointed out that the incomplete Cholesky factorization algorithm can be used to yield a low-rank approximation to $K$. There are many other papers on approximate methods for kernel machines, but as these do not focus on low-rank approximations of $K$ references are omitted due to space limitations.

## 1.4 Fast approximate Gaussian process classification and regression

The technique described above is potentially applicable to a wide range of kernel methods, including Gaussian process (GP) and Support Vector machine algorithms. We specialize here on GP regression and classification.

GP regression requires us to solve the system $(K + \sigma I)\boldsymbol{a} = \boldsymbol{t}$ where $\boldsymbol{t}$ are the training targets. The Bayes predictor is then given by $\hat{y}(\boldsymbol{x}) = \boldsymbol{a}^T \boldsymbol{k}(\boldsymbol{x})$ where $\boldsymbol{k}(\boldsymbol{x})$ is the vector of length $n$ with elements $k(\boldsymbol{x}_i, \boldsymbol{x})$. Replacing the Gram matrix in the linear system by the approximation $\tilde{K}$, the system can easily be solved using the *Woodbury formula* (see, e.g. Press et al., 1992):

$$\boldsymbol{a} = \frac{1}{\sigma}\left(\boldsymbol{t} - \tilde{U}\left(\sigma I + \tilde{\Lambda}\tilde{U}^T\tilde{U}\right)^{-1}\tilde{\Lambda}\tilde{U}^T\boldsymbol{t}\right) \tag{11}$$

which is $O(p^2 n)$. The memory cost is $O(np)$.

The posterior for the Gaussian process classification model cannot be computed in a tractable way, but an approximation based on the *Laplace approximation* has been proposed which works well in practice (e.g. see Williams and Barber (1998)). The method is iterative and requires for each iteration the solution of a system of the form

$$(I + W(K + \sigma I))\boldsymbol{a} = W\boldsymbol{y} + \boldsymbol{\pi} \tag{12}$$

to be computed, where $\boldsymbol{y} = (K + \sigma I)\boldsymbol{a}_{old}$, $W$ is a diagonal matrix and $\boldsymbol{\pi}$ and $W$ depend on $\boldsymbol{y}$. Using the approximation $\tilde{K}$ this can be computed efficiently as follows. Let $A = \tilde{\Lambda}\tilde{U}^T$. In each iteration, we compute $D = I + \sigma W$, $\boldsymbol{b} = D^{-1}(W\boldsymbol{y} + \boldsymbol{\pi})$ and $B = D^{-1}W\tilde{U}$. Then we have by Woodbury's formula:

$$\boldsymbol{a} = \boldsymbol{b} - B\left(I + AB\right)^{-1}A\boldsymbol{b}. \tag{13}$$

Each iteration is therefore $O(p^2 n)$. The memory cost is again $O(np)$.

A formula for the computation of determinants analogous to the Woodbury formula for matrix inversion is also available. This allows the marginal likelihood $P(\boldsymbol{t}|\theta)$ to be approximated efficiently, and used as a means of selecting the kernel parameters $\theta$. (Williams and Barber (1998) describe how to approximate the marginal likelihood in the classification case.) Of course other methods like cross-validation could also be used for kernel optimization.

## 2 Experimental Results

We present experimental results on classification and regression tasks.

### 2.1 Classification Experiments

Our experiments were carried out using the US Postal Service (USPS) handwritten digit database. Each example has 256 inputs, being the scaled grey-level values of

the $16 \times 16$ pixels. There are 7291 training examples and 2007 test examples. There are ten different output classes, corresponding to the digits $0, \ldots, 9$.

Gaussian process classifiers were trained using the Laplace approximation as described in Williams and Barber (1998). The kernel was of the form $k(\boldsymbol{x}, \boldsymbol{y}) = v_0 \exp(- \parallel \boldsymbol{x} - \boldsymbol{y} \parallel^2 /(0.5 \cdot 16^2))$ where the width 0.5 equals twice the average of the data variance on each dimension, as used in Schölkopf et al. (1999). The scaling factor $v_0$ was set to the reasonable value of 10. The jitter factor $\sigma$ was set to $10^{-6}$. We describe three experiments[1].

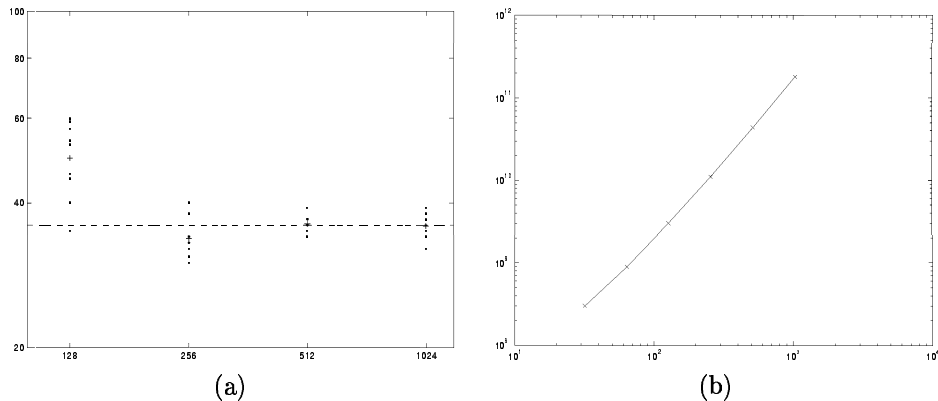

Figure 1: (a) log-log plot of the number of errors against $m$. Dashed line shows performance of the classifier using the full GP classifier, the $+$ symbol denotes the mean of the 10 results for each $m$. (b) log-log plot of number of flops against $m$.

**Experiment 1**. In the first experiment, the task is to discriminate the digits of class "4" from the rest. We vary $m$, the size of the subset used, and for each value of $m$ use 10 different subsets of training data of size $m$ so as to be able to assess the effect of this variability on the results. These results are compared to a score of 36 errors (out of 2007) for the full GP classifier (i.e. without approximating $K$). In Figure 1(a) we plot (on a log-log scale) the results for $m = 1024, 512, 256$ and 128. Good performance is obtained down to $m = 256$, although it deteriorates for $m = 128$. Good performance can also be obtained using $p < m$; for example with $m = 1024$ good results are obtained with $p = 512, 256$ and 128. In Figure 1(b) the number of flops taken to compute the MAP value of $\boldsymbol{a}$ (as defined in equation 12) is plotted against $m$; this verifies the $O(m^2)$ scaling behaviour of the computation.

**Experiment 2**. The fact that the best performance obtained is close to (and sometimes better than) that of the full GP classifier using all 7291 training examples suggests that all of the information in the training set is being utilized, not just the targets corresponding $m$ training points. To examine this issue in more detail we compared the performance of a full GP classifier using only a subset of the training data of size $m$, against the Nyström approximation (with $p = m$) which computes a $m \times m$ matrix eigendecomposition, but makes use of all $n$ training data points. Again for each $m$ we used 10 different samples from the $n$ training examples (sampled without replacement), and performed paired comparisons. The task was to discriminate '4's from the rest of the digits, as in the first experiment.

The mean and standard deviation for the performance of each classifier for $m =$

| $m$ | 1024 | 512 | 256 | 128 | 64 |
|---|---|---|---|---|---|
| Ny mean | 35.9 | 34.7 | 34.5 | 46.8 | 101.3 |
| Ny std dev | 1.97 | 2.54 | 2.99 | 6.89 | 22.92 |
| GP mean | 54.1 | 64.6 | 77.2 | 102.9 | 127.4 |
| GP std dev | 4.48 | 6.28 | 13.16 | 25.01 | 28.47 |
| Diff mean | 18.2 | 29.9 | 42.7 | 56.1 | 26.1 |
| $t$-statistic | 11.02 | 12.20 | 9.00 | 6.37 | 3.41 |

Table 1: Performance of the full GP classifier using only $m$ data points (denoted GP) compared with the Nyström classifier using a eigendecomposition of size $m$ (denoted Ny), for $m = 1024, 512, 256, 128, 64$. The mean and standard deviation of the number of errors is reported for both classifiers, along with the mean and $t$-statistic computed from the paired differences.

1024, 512, 256, 128, 64 are given in Table 1, along with mean of the differences and the $t$-statistic computed from the paired differences. We see that the mean performance is always better for the Nyström classifier. As $t_{0.025,9} = 2.26$ all five results are significant at the 5% level. Note also that the variance due to different samples of size $m$ is smaller for the Nyström classifier than for the full GP classifier.

| | 0 | 1 | 2 | 3 | 4 | 5 | 6 | 7 | 8 | 9 |
|---|---|---|---|---|---|---|---|---|---|---|
| GP(7291) | 18 | 15 | 37 | 33 | 36 | 33 | 14 | 15 | 36 | 24 |
| Eig(256) | 19 | 15 | 35 | 30 | 36 | 28 | 13 | 15 | 33 | 21 |
| Ny(256,1024) | 15 | 15 | 37 | 36 | 37 | 29 | 14 | 14 | 33 | 20 |
| Ny(256, 512) | 22 | 14 | 35 | 32 | 32 | 27 | 14 | 13 | 36 | 24 |
| Ny(256,256) | 18 | 15 | 33 | 31 | 26 | 33 | 12 | 18 | 43 | 31 |

Table 2: Comparison of the errors rates of the GP(7291), Eig(256), Ny(256,1024), Ny(256,512) and Ny(256,256) classifiers for the 10 different tasks.

**Experiment 3**. The third experiment considers the results obtained for the ten different tasks of discriminating one digit from all of the others. In Table 2 we give the number of errors for four different predictors (a) GP(7291), the full GP classifier; (b) Eig(256), the GP predictor using an exact eigendecomposition of the full $K$ matrix and retaining $p = 256$ eigenvalues (c) Ny(256,1024) the Nyström predictor using $p = 256$, $m = 1024$ (d) Ny(256,512) and (e) Ny(256,256). The results in most cases are very similar. Notice that the results for Eig(256) and Ny(256,512) are the same as or better than GP(7291) for 9 out of the 10 cases, the results of Ny(256,1024) are the same as or better than GP(7291) in 8 out of 10 cases and the results of Ny(256,256) are the same as or better than GP(7291) in 7 out of 10 cases.

We have also implemented an equal-weight committee of up to 10 Ny$(p,m)$ predictors for small $p$ and $m$, but have not found improvements over a single predictor of the same size.

## 2.2 Regression Experiments

We have also tested the Nyström method on the `abalone` regression problem, taken from the UCI repository `http://www.ics.uci.edu/~mlearn/MLRepository.html`, as used by Smola and Schölkopf (2000). The problem has 8 input variables (scaled to zero mean and unit variance), 3133 training examples and 1044 test examples. We

used a Gaussian kernel with the same parameter settings as Smola and Schölkopf. Excellent agreement with the exact method was obtained for $m = 1000$, 500 and 250, but performance declined quite markedly for $m = 125$.

## 3    Discussion

We have seen above that the Nyström approximation can allow a very significant speed-up of the computations required for the GP classifier without sacrificing accuracy. This speed-up comes about from the insight that matrix eigenproblems of different dimensions are related because they are all approximations of the eigenfunction equation 2. An advantage of the method is that it is not necessary to compute or store the whole Gram matrix, but only a $m \times n$ portion of it. For problems with thousands of examples we have shown that good performance can be obtained using values of $m$ of only a few hundred. As $n$ increases, we would expect that the ratio $m/n$ could be made even smaller. The approximation is likely to be particularly good for kernels (like the Gaussian kernel) for which the eigenvalues decay rapidly.

### Acknowledgements

We thank Amos Storkey for helpful discussions and the anonymous NIPS referees who helped improve this paper. MS gratefully acknowledges support through a research studentship from Microsoft Research Ltd.

## Footnotes

[1]In these classification experiments each $\tilde{\boldsymbol{u}}_i^{(n)}$ (see equation 9) was normalized to have length 1.

## References

Baker, C. T. H. (1977). *The numerical treatment of integral equations.* Oxford: Clarendon Press.

Fine, S., & Scheinberg, K. (2000). *Efficient SVM Training Using Low-Rank Kernel Representation*Research Report RC 21911). IBM T. J. Watson Research Center.

Frieze, A., Kannan, R., & Vempala, S. (1998). Fast Monte-Carlo Algorithms for finding low-rank approximations. *39th Conference on the Foundations of Computer Science* (pp. 370–378).

Neal, R. M. (1998). Regression and classification using Gaussian process priors (with discussion). In J. M. Bernardo et al. (Eds.), *Bayesian statistics 6*, 475–501. Oxford University Press.

Press, W. H., Teukolsky, S. A., Vetterling, W. T., & Flannery, B. P. (1992). *Numerical Recipes in C.* Cambridge University Press. Second edition.

Schölkopf, B., Mika, S., Burges, C. J. C., et al. (1999). Input space vs feature space in kernel-based methods. *IEEE Transactions on Neural Networks, 10(5)*, 1000–1017.

Schölkopf, B., Smola, A., & Müller, K.-R. (1998). Nonlinear component analysis as a kernel eigenvalue problem. *Neural Computation, 10*, 1299–1319.

Smola, A. J., & Schölkopf, B. (2000). Sparse Greedy Matrix Approximation for Machine Learning. *Proceedings of the Seventeenth International Conference on Machine Learning.* Morgan Kaufmann.

Vapnik, V. N. (1995). *The nature of statistical learning theory.* New York: Springer Verlag.

Wahba, G. (1990). *Spline models for observational data.* Philadelphia, PA: Society for Industrial and Applied Mathematics. CBMS-NSF Regional Conference series in applied mathematics.

Williams, C. K. I., & Barber, D. (1998). Bayesian classification with Gaussian processes. *IEEE Transactions on Pattern Analysis and Machine Intelligence, 20(12)*, 1342–1351.
